# How memory biases affect information transmission: A rational analysis of serial reproduction

**Jing Xu   Thomas L. Griffiths**
Department of Psychology
University of California, Berkeley
Berkeley, CA 94720-1650
{jing.xu,tom_griffiths}@berkeley.edu

## Abstract

Many human interactions involve pieces of information being passed from one person to another, raising the question of how this process of information transmission is affected by the capacities of the agents involved. In the 1930s, Sir Frederic Bartlett explored the influence of memory biases in "serial reproduction" of information, in which one person's reconstruction of a stimulus from memory becomes the stimulus seen by the next person. These experiments were done using relatively uncontrolled stimuli such as pictures and stories, but suggested that serial reproduction would transform information in a way that reflected the biases inherent in memory. We formally analyze serial reproduction using a Bayesian model of reconstruction from memory, giving a general result characterizing the effect of memory biases on information transmission. We then test the predictions of this account in two experiments using simple one-dimensional stimuli. Our results provide theoretical and empirical justification for the idea that serial reproduction reflects memory biases.

## 1 Introduction

Most of the facts that we know about the world are not learned through first-hand experience, but are the result of information being passed from one person to another. This raises a natural question: how are such processes of information transmission affected by the capacities of the agents involved? Decades of memory research have charted the ways in which our memories distort reality, changing the details of experiences and introducing events that never occurred (see [1] for an overview). We might thus expect that these memory biases would affect the transmission of information, since such a process relies on each person remembering a fact accurately.

The question of how memory biases affect information transmission was first investigated in detail in Sir Frederic Bartlett's "serial reproduction" experiments [2]. Bartlett interpreted these studies as showing that people were biased by their own culture when they reconstruct information from memory, and that this bias became exaggerated through serial reproduction. Serial reproduction has become one of the standard methods used to simulate the process of cultural transmission, and several subsequent studies have used this paradigm (e.g., [3, 4]). However, this phenomenon has not been systematically and formally analyzed, and most of these studies have used complex stimuli that are semantically rich but hard to control. In this paper, we formally analyze and empirically evaluate how information is changed by serial reproduction and how this process relates to memory biases. In particular, we provide a rational analysis of serial reproduction (in the spirit of [5]), considering how information should change when passed along a chain of rational agents.

Biased reconstructions are found in many tasks. For example, people are biased by their knowledge of the structure of categories when they reconstruct simple stimuli from memory. One common

effect of this kind is that people judge stimuli that cross boundaries of two different categories to be further apart than those within the same category, although the distances between the stimuli are the same in the two situations [6]. However, biases need not reflect suboptimal performance. If we assume that memory is solving the problem of extracting and storing information from the noisy signal presented to our senses, we can analyze the process of reconstruction from memory as a Bayesian inference. Under this view, reconstructions should combine prior knowledge about the world with the information provided by noisy stimuli. Use of prior knowledge will result in biases, but these biases ultimately make memory more accurate [7].

If this account of reconstruction from memory is true, we would expect the same inference process to occur at every step of serial reproduction. The effects of memory biases should thus be accumulated. Assuming all participants share the same prior knowledge about the world, serial reproduction should ultimately reveal the nature of this knowledge. Drawing on recent work exploring other processes of information transmission [8, 9], we show that a rational analysis of serial reproduction makes exactly this prediction. To test the predictions of this account, we explore the special case where the task is to reconstruct a one-dimensional stimulus using the information that it is drawn from a fixed Gaussian distribution. In this case we can precisely characterize behavior at every step of serial reproduction. Specifically, we show that this defines a simple first-order autoregressive, or AR(1), process, allowing us to draw on a variety of results characterizing such processes. We use these predictions to test the Bayesian models of serial reproduction in two laboratory experiments and show that the predictions hold serial reproduction both between- and within-subjects.

The plan of the paper is as follows. Section 2 lays out the Bayesian account of serial reproduction. In Section 3 we show how this Bayesian account corresponds to the AR(1) process. Sections 4 and 5 present two experiments testing the model's prediction that serial reproduction reveals memory biases. Section 6 concludes the paper.

## 2 A Bayesian view of serial reproduction

We will outline our Bayesian approach to serial reproduction by first considering the problem of reconstruction from memory, and then asking what happens when the solution to this problem is repeated many times, as in serial reproduction.

### 2.1 Reconstruction from memory

Our goal is to give a rational account of reconstruction from memory, considering the underlying computational problem and finding the optimal solution to that problem. We will formulate the problem of reconstruction from memory as a problem of inferring and storing accurate information about the world from noisy sensory data. Given a noisy stimulus $x$, we seek to recover the true state of the world $\mu$ that generated that stimulus, storing an estimate $\hat{\mu}$ in memory. The optimal solution to this problem is provided by Bayesian statistics. Previous experience provides a "prior" distribution on possible states of the world, $p(\mu)$. On observing $x$, this can be updated to a "posterior" distribution $p(\mu|x)$ by applying Bayes' rule

$$p(\mu|x) = \frac{p(x|\mu)p(\mu)}{\int p(x|\mu)p(\mu)\,d\mu} \tag{1}$$

where $p(x|\mu)$ – the "likelihood" – indicates the probability of observing $x$ if $\mu$ is the true state of the world. Having computed $p(\mu|x)$, a number of schemes could be used to select an estimate of $\hat{\mu}$ to store. Perhaps the simplest such scheme is sampling from the posterior, with $\hat{\mu} \sim p(\mu|x)$.

This analysis provides a general schema for modeling reconstruction from memory, applicable for any form of $x$ and $\mu$. A simple example is the special case where $x$ and $\mu$ vary along a single continuous dimension. In the experiment presented later in the paper we take this dimension to be the width of a fish, showing people a fish and asking them to reconstruct its width from memory, but the dimension of interest could be any subjective quantity such as the perceived length, loudness, duration, or brightness of a stimulus. Assume that previous experience establishes that $\mu$ has a Gaussian distribution, with $\mu \sim N(\mu_0, \sigma_0^2)$, and that the noise process means that $x$ has a Gaussian distribution centered on $\mu$, $x|\mu \sim N(\mu, \sigma_x^2)$. In this case, we can use standard results from Bayesian statistics [10] to show that the outcome of Equation 1 is also a Gaussian distribution, with $p(\mu|x)$ being $N(\lambda x + (1 - \lambda)\mu_0, \lambda\sigma_x^2)$, where $\lambda = 1/(1 + \sigma_x^2/\sigma_0^2)$.

The analysis presented in the previous paragraph makes a clear prediction: that the reconstruction $\hat{\mu}$ should be a compromise between the observed value $x$ and the mean of the prior $\mu_0$, with the terms of the compromise being set by the ratio of the noise in the data $\sigma_x^2$ to the uncertainty in the prior $\sigma_0^2$. This model thus predicts a systematic bias in reconstruction that is not a consequence of an error of memory, but the optimal solution to the problem of extracting information from a noisy stimulus. Huttenlocher and colleagues [7] have conducted several experiments testing this account of memory biases, showing that people's reconstructions interpolate between observed stimuli and the mean of a trained distribution as predicted. Using a similar notion of recosntruction from memory, Hemmer and Steyvers [11] have conducted experiments to show that people formed appropriate Bayesian reconstructions for realistic stimuli such as images of fruit, and seemed capable of drawing on prior knowledge at multiple levels of abstraction in doing so.

## 2.2 Serial reproduction

With a model of how people might approach the problem of reconstruction from memory in hand, we are now in a position to analyze what happens in serial reproduction, where the stimuli that people receive on one trial are the results of a previous reconstruction. On the $n$th trial, a participant sees a stimulus $x_n$. The participant then computes $p(\mu|x_n)$ as outlined in the previous section, and stores a sample $\hat{\mu}$ from this distribution in memory. When asked to produce a reconstruction, the participant generates a new value $x_{n+1}$ from a distribution that depends on $\hat{\mu}$. If the likelihood, $p(x|\mu)$, reflects perceptual noise, then it is reasonable to assume that $x_{n+1}$ will be sampled from this distribution, substituting $\hat{\mu}$ for $\mu$. This value of $x_{n+1}$ is the stimulus for the next trial.

Viewed from this perspective, serial reproduction defines a stochastic process: a sequence of random variables evolving over time. In particular, it is a Markov chain, since the reconstruction produced on the current trial depends only on the value produced on the preceding trial (e.g. [12]). The transition probabilities of this Markov chain are

$$p(x_{n+1}|x_n) = \int p(x_{n+1}|\mu)p(\mu|x_n)\,d\mu \tag{2}$$

being the probability that $x_{n+1}$ is produced as a reconstruction for the stimulus $x_n$. If this Markov chain is ergodic (see [12] for details) it will converge to a stationary distribution $\pi(x)$, with $p(x_n|x_1)$ tending to $\pi(x_n)$ as $n \to \infty$. That is, after many reproductions, we should expect the probability of seeing a particular stimulus being produced as a reproduction to stabilize to a fixed distribution. Identifying this distribution will help us understand the consequences of serial reproduction.

The transition probabilities given in Equation 2 have a special form, being the result of sampling a value from the posterior distribution $p(\mu|x_n)$ and then sampling a value from the likelihood $p(x_{n+1}|\mu)$. In this case, it is possible to identify the stationary distribution of the Markov chain [8, 9]. The stationary distribution of this Markov chain is the *prior predictive distribution*

$$\pi(x) = \int p(x|\mu)p(\mu)\,d\mu \tag{3}$$

being the probability of observing the stimulus $x$ when $\mu$ is sampled from the prior. This happens because this Markov chain is a Gibbs sampler for the joint distribution on $x$ and $\mu$ defined by multiplying $p(x|\mu)$ and $p(\mu)$ [9]. This gives a clear characterization of the consequences of serial reproduction: after many reproductions, the stimuli being produced will be sampled from the prior distribution assumed by the participants. Convergence to the prior predictive distribution provides a formal justification for the traditional claims that serial reproduction reveals cultural biases, since those biases would be reflected in the prior.

In the special case of reconstruction of stimuli that vary along a single dimension, we can also analytically compute the probability density functions for the transition probabilities and stationary distribution. Applying Equation 2 using the results summarized in the previous section, we have $x_{n+1}|x_n \sim N(\mu_n, (\sigma_x^2 + \sigma_n^2))$, where $\mu_n = \lambda x_n + (1-\lambda)\mu_0$, and $\sigma_n^2 = \lambda\sigma_x^2$. Likewise, Equation 3 indicates that the stationary distribution is $N(\mu_0, (\sigma_x^2 + \sigma_0^2))$. The rate at which the Markov chain converges to the stationary distribution depends on the value of $\lambda$. When $\lambda$ is close to 1, convergence is slow since $\mu_n$ is close to $x_n$. As $\lambda$ gets closer to 0, $\mu_n$ is more influenced by $\mu_0$ and convergence is faster. Since $\lambda = 1/(1 + \sigma_x^2/\sigma_0^2)$, the convergence rate thus depends on the ratio of the participant's perceptual noise and the variance of the prior distribution, $\sigma_x^2/\sigma_0^2$. More perceptual noise results in

faster convergence, since the specific value of $x_n$ is trusted less; while more uncertainty in the prior results in slower convergence, since $x_n$ is given greater weight.

## 3    Serial reproduction of one-dimensional stimuli as an AR(1) process

The special case of serial reproduction of one-dimensional stimuli can also give us further insight into the consequences of modifying our assumptions about storage and reconstruction from memory, by exploiting a further property of the underlying stochastic process: that it is a first-order autoregressive process, abbreviated to AR(1). The general form of an AR(1) process is

$$x_{n+1} = c + \phi x_n + \epsilon_{n+1} \tag{4}$$

where $\epsilon_{n+1} \sim N(0, \sigma_\epsilon^2)$. Equation 4 has the familiar form of a regression equation, predicting one variable as a linear function of another, plus Gaussian noise. It defines a stochastic process because each variable is being predicted from that which precedes it in sequence. AR(1) models are widely used to model timeseries data, being one of the simplest models for capturing temporal dependency.

Just as showing that a stochastic process is a Markov chain provides information about its dynamics and asymptotic behavior, showing that it reduces to an AR(1) process provides access to a number of results characterizing the properties of these processes. If $\phi < 1$ the process has a stationary distribution that is Gaussian with mean $c/(1 - \phi)$ and variance $\sigma_\epsilon^2/(1 - \phi^2)$. The autocovariance at a lag of $n$ is $\phi^n \sigma_\epsilon^2/(1 - \phi^2)$, and thus decays geometrically in $\phi$. An AR(1) process thus converges to its stationary distribution at a rate determined by $\phi$.

It is straightforward to show that the stochastic process defined by serial reproduction where a sample from the posterior distribution on $\mu$ is stored in memory and a new value $x$ is sampled from the likelihood is an AR(1) process. Using the results in the previous section, at the $(n + 1)$th iteration

$$x_{n+1} = (1 - \lambda)\mu_0 + \lambda x_n + \epsilon_{n+1} \tag{5}$$

where $\lambda = 1/(1 + \sigma_x^2/\sigma_0^2)$ and $\epsilon_{n+1} \sim N(0, (\sigma_x^2 + \sigma_n^2))$ with $\sigma_n^2 = \lambda \sigma_x^2$. This is an AR(1) process with $c = (1 - \lambda)\mu_0$, $\phi = \lambda$, and $\sigma_\epsilon^2 = \sigma_x^2 + \sigma_n^2$. Since $\lambda$ is less than 1 for any $\sigma_0^2$ and $\sigma_x^2$, we can find the stationary distribution by substituting these values into the expressions given above.

Identifying serial reproduction for single-dimensional stimuli as an AR(1) process allows us to relax our assumptions about the way that people are storing and reconstructing information. The AR(1) model can accommodate different assumptions about memory storage and reconstruction.[1] All these ways of characterizing serial reproduction lead to the same basic prediction: that repeatedly reconstructing stimuli from memory will result in convergence to a distribution whose mean corresponds to the mean of the prior. In the remainder of the paper we test this prediction.

In the following sections, we present two serial reproduction experiments conducted with stimuli that vary along only one dimension (width of fish). The first experiment follows previous research in using a between-subjects design, with the reconstructions of one participant serving as the stimuli for the next. The second experiment uses a within-subjects design in which each person reconstructs stimuli that they themselves produced on a previous trial, testing the potential of this design to reveal the memory biases of individuals.

## 4    Experiment 1: Between-subjects serial reproduction

This experiment directly tested the basic prediction that the outcome of serial reproduction will reflect people's priors. Two groups of participants were trained on different distributions of a one-dimensional quantity – the width of a schematic fish – that would serve as a prior for reconstructing

similar stimuli from memory. The two distributions differed in their means, allowing us to examine whether the mean of the distribution produced by serial reproduction is affected by the prior.

## 4.1 Method

The experiment followed the same basic procedure as Bartlett's classic experiments [2]. Participants were 46 members of the university community. Stimuli were the same as those used in [7]: fish with elliptical bodies and fan-shaped tails. All the fish stimuli varied only in one dimension, the width of the fish, ranging from 2.63cm to 5.76cm. The stimuli were presented on an Apple iMac computer by a Matlab script using PsychToolBox extensions [13, 14].

Participants were first trained to discriminate fish-farm and ocean fish. The width of the fish-farm fish was normally distributed and that of the ocean fish was uniformly distributed between 2.63 and 5.75cm. Two groups of participants were trained on one of the two distributions of fish-farm fish (prior distributions A and B), with different means and same standard deviations. In condition A, $\mu_0 = 3.66$cm, $\sigma_0 = 1.3$cm; in condition B, $\mu_0 = 4.72$cm, $\sigma_0 = 1.3$cm.

In the training phase, participants first received a block of 60 trials. On each trial, a stimulus was presented at the center of a computer monitor and participants tried to predict which type of fish it was by pressing one of the keys on the keyboard and they received feedback about the correctness of the prediction. The participants were then tested for 20 trials on their knowledge of the two types of fish. The procedure was the same as the training block except there was no feedback. The training-testing loop was repeated until the participants reached 80% correct in using the optimal decision strategy. If a participant could not pass the test after five iterations, the experiment halted.

In the reproduction phase, the participants were told that they were to record fish sizes for the fish farm. On each trial, a fish stimulus was flashed at the center of the screen for 500ms and then disappeared. Another fish of random size appeared at one of four possible positions near the center of screen and the participants used the up and down arrow keys to adjust the width of the fish until they thought it matched the fish they just saw. The fish widths seen by the first participant in each condition were 120 values randomly sampled from a uniform distribution from 2.63 to 5.75cm. The first participant tried to memorize these random samples and then gave the reconstructions. Each subsequent participant in each condition was then presented with the data generated by the previous participant and they again tried to reconstruct those fish widths. Thus, each participant's data constitute one slice of time in 120 serial reproduction chains.

At the end of the experiment, the participants were given a final 50-trial test to check if their prior distributions had drifted. Ten participants' data were excluded from the chains based on three criteria: 1) final testing score was less than 80% of optimal performance; 2) the difference between the reproduced value and stimulus shown was greater than the difference between the largest and the smallest stimuli in the training distribution on any trial; 3) there were no adjustments from the starting value of the fish width for more than half of the trials.

## 4.2 Results and Discussion

There were 18 participants in each condition, resulting in 18 generations of serial reproduction. Figure 1 shows the initial and final distributions of the reconstructions, together with the autoregression plots for the two conditions. The mean reconstructed fish widths produced by the first participants in conditions A and B were $4.22$ and $4.21$cm respectively, which were not statistically significantly different ($t(238) = 0.09$, $p = 0.93$). For the final participants in each chain, the mean reconstructed fish widths were $3.20$ and $3.68$cm respectively, a statistically significant difference ($t(238) = 6.93$, $p < 0.001$). The difference in means matches the direction of the difference in the training provided in conditions A and B, although the overall size of the difference is reduced and the means of the stationary distributions were lower than those of the distributions used in training.

The autoregression plots provide a further quantitative test of the predictions of our Bayesian model. The basic prediction of the model is that reconstruction should look like regression, and this is exactly what we see in Figure 1. The correlation between the stimulus $x_n$ and its reconstruction $x_{n+1}$ is the correlation between the AR(1) model's predictions and the data, and this correlation was high in both conditions, being 0.91 and 0.86 ($p < 0.001$) for conditions A and B respectively. Finally, we examined whether the Markov assumption underlying our analysis was valid, by computing the

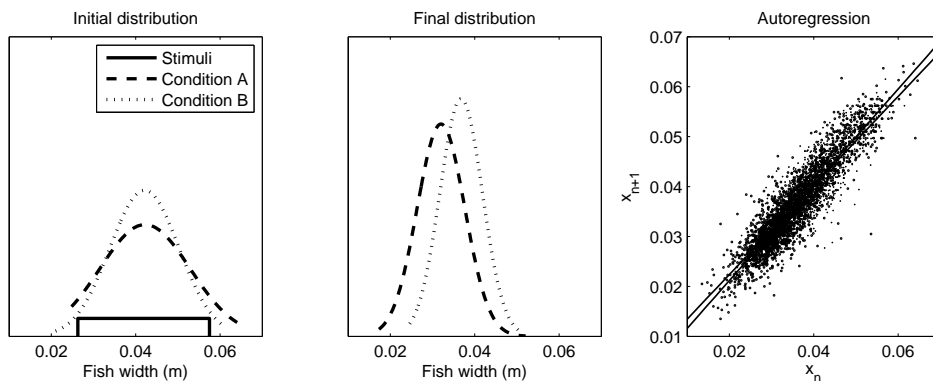

Figure 1: Initial and final distributions for the two conditions in Experiment 1. (a) The distribution of stimuli and Gaussian fits to reconstructions for the first participants in the two conditions. (b) Gaussian fits to reconstructions generated by the 18th participants in each condition. (c) Autoregression plot for $x_{n+1}$ as a function of $x_n$ for the two conditions.

correlation between $x_{n+1}$ and $x_{n-1}$ given $x_n$. The resulting partial correlation was low for both conditions, being 0.04 and 0.01 in conditions A and B respectively (both $p < 0.05$).

## 5  Experiment 2: Within-subjects serial reproduction

The between-subjects design allows us to reproduce the process of information transmission, but our analysis suggests that serial reproduction might also have promise as a method for investigating the memory biases of individuals. To explore the potential of this method, we tested the model with a within-subjects design, in which a participant's reproduction in the current trial became the stimulus for that same participant in a later trial. Each participant's responses over the entire experiment thus produced a chain of reproductions. Each participant produced three such chains, starting from widely separated initial values. Control trials and careful instructions were used so that the participants would not realize that some of the stimuli were their own reproductions.

### 5.1  Method

Forty-six undergraduates from the university research participation pool participated the experiment. The basic procedure was the same as Experiment 1, except in the reproduction phase. Each participant's responses in this phase formed three chains of 40 trials. The chains started with three original stimuli with width values of 2.63cm, 4.19cm, and 5.76cm, then in the following trials, the stimuli participants saw were their own reproductions in the previous trials in the same chain. To prevent participants from realizing this fact, chain order was randomized and the Markov chain trials were intermixed with 40 control trials in which widths were drawn from the prior distribution.

### 5.2  Results and Discussion

Participants' data were excluded based on the same criteria as used in Experiment 1, with a lower testing score of 70% of optimal performance and one additional criterion relevant to the within-subjects case: participants were also excluded if the three chains did not converge, with the criterion for convergence being that the lower and upper chains must cross the middle chain. After these screening procedures, 40 participants' data were accepted, with 21 in condition A and 19 in condition B. It took most participants about 20 trials for the chains to converge, so only the second half of the chains (trials 21-40) were analyzed further.

The locations of the stationary distributions were measured by computing the means of the reproduced fish widths for each participant. For conditions A (3.66cm) and B (4.72cm), the average of these means was 3.32 and 4.01cm respectively ($t(38) = 2.41$, $p = 0.021$). The right panel of Figure

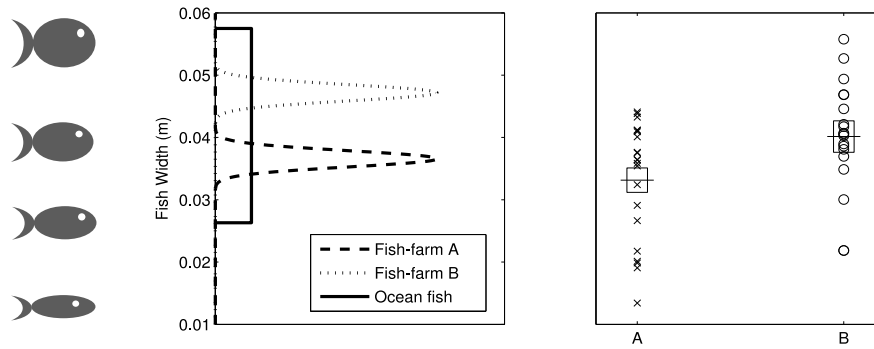

Figure 2: Stimuli, training distributions and stationary distributions for Experiment 2. Each data point in the right panel shows the mean of the last 20 iterations for a single participant. Boxes show the 95% confidence interval around the mean for each condition.

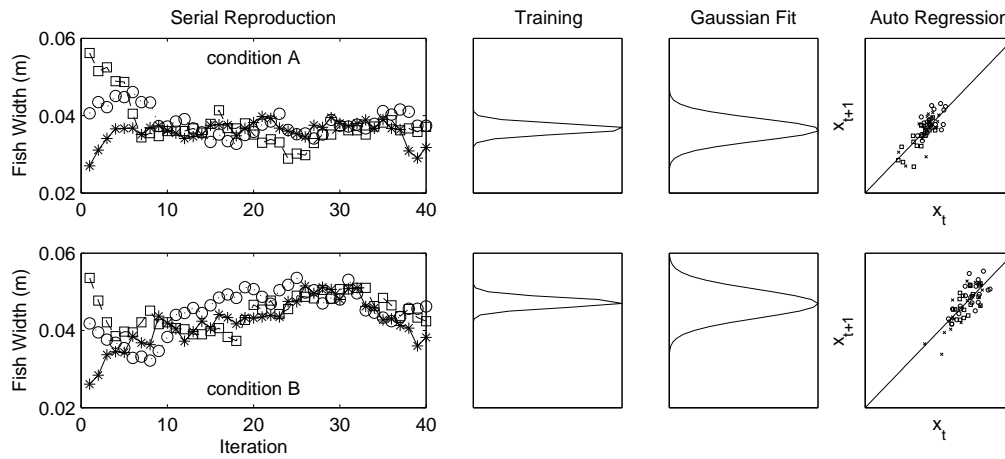

Figure 3: Chains and stationary distributions for individual participants from the two conditions. (a) The three Markov chains generated by each participant, starting from three different values. (b) Training distributions for each condition. (c) Gaussian fits for the last 20 iterations of each participant's data. (d) Autoregression for the last 20 iterations of each participant's data.

2 shows the mean values for these two conditions. The basic prediction of the model was borne out: participants converged to distributions that differed significantly in their means when they were exposed to data suggesting a different prior. However, the means were in general lower than those of the prior. This effect was less prominent in the control trials, which produced means of 3.63 and 4.53cm respectively.[2]

Figure 3 shows the chains, training distributions, the Gaussian fits and the autoregression for the second half of the Markov chains for two participants in the two conditions. Correlation analysis showed that the AR(1) model's predictions are highly correlated with the data generated by each participant, with mean correlations being 0.90 and 0.81 for conditions A and B respectively. The

correlations are significant for all participants. The mean partial correlation between $x_{t+1}$ and $x_{t-1}$ given $x_t$ was low, being 0.07 and 0.11 for conditions A and B respectively, suggesting that the Markov assumption was satisfied. The partial correlations were significant ($p < 0.05$) for only one participant in condition B.

## 6    Conclusion

We have presented a Bayesian account of serial reproduction, and tested the basic predictions of this account using two strictly controlled laboratory experiments. The results of these experiments are consistent with the predictions of our account, with serial reproduction converging to a distribution that is influenced by the prior distribution established through training. Our analysis connects the biases revealed by serial reproduction with the more general Bayesian strategy of combining prior knowledge with noisy data to achieve higher accuracy [7]. It also shows that serial reproduction can be analyzed using Markov chains and first-order autoregressive models, providing the opportunity to draw on a rich body of work on the dynamics and asymptotic behavior of such processes. These connections allows us to provide a formal justification for the idea that serial reproduction changes the information being transmitted in a way that reflects the biases of the people transmitting it, establishing that this result holds under several different characterizations of the processes involved in storage and reconstruction from memory.

**Acknowledgments**

This work was supported by grant number 0704034 from the National Science Foundation.

## Footnotes

[1]In the memorization phase, the participant's memory $\hat{\mu}$ can be 1) a sample from the posterior distribution $p(\mu|x_n)$, as assumed above, or 2) a value such that $\hat{\mu} = \mathrm{argmax}_\mu \, p(\mu|x_n)$, which is also the expected value of the Gaussian posterior, $p(\mu|x_n)$. In the reproduction phase, the participant's reproduction $x_{n+1}$ can be 1) a noisy reconstruction, which is a sample from the likelihood $p(x_{n+1}|\hat{\mu})$, as assumed above, or 2) a perfect reconstruction from memory, such that $x_{n+1} = \hat{\mu}$. This defines four different models of serial reproduction, all of which correspond to AR(1) processes that differ only in the variance $\sigma_\epsilon^2$ (although maximizing $p(\mu|x_n)$ and then storing a perfect reconstruction is degenerate, with $\sigma_\epsilon^2 = 0$). In all four cases serial reproduction thus converges to a Gaussian stationary distribution with mean $\mu_0$, but with different variances.

[2]Since both experiments produced stationary distributions with means lower than those of the training distributions, we conducted a separate experiment examining the reconstructions that people produced without training. The mean fish width produced by 20 participants was 3.43cm, significantly less than the mean of the initial values of each chain, 4.19cm ($t(19) = 3.75$, $p < 0.01$). This result suggested that people seem to have an *a priori* expectation that fish will have widths smaller than those used as our category means, suggesting that people in the experiments are using a prior that is a compromise between this expectation and the training data.

## References

[1] D. L. Schacter, J. T. Coyle, G. D. Fischbach, M. M. Mesulam, and L. E. Sullivan, editors. *Memory distortion: How minds, brains, and societies reconstruct the past*. Harvard University Press, Cambridge, MA, 1995.

[2] F. C. Bartlett. *Remembering: a study in experimental and social psychology*. Cambridge University Press, Cambridge, 1932.

[3] A. Bangerter. Transformation between scientific and social representations of conception: The method of serial reproduction. *British Journal of Social Psychology*, 39:521–535, 2000.

[4] J. Barrett and M. Nyhof. Spreading nonnatural concepts: The role of intuitive conceptual structures in memory and transmission of cultural materials. *Journal of Cognition and Culture*, 1:69–100, 2001.

[5] J. R. Anderson. *The adaptive character of thought*. Erlbaum, Hillsdale, NJ, 1990.

[6] A. M. Liberman, F. S. Cooper, D. P. shankweiler, and M. Studdert-Kennedy. Perception of the speech code. *Psychological Review*, 74:431–461, 1967.

[7] J. Huttenlocher, L. V. Hedges, and J. L. Vevea. Why do categories affect stimulus judgment? *Journal of Experimental Psychology: General*, pages 220–241, 2000.

[8] T. L. Griffiths and M. L. Kalish. A Bayesian view of language evolution by iterated learning. In B. G. Bara, L. Barsalou, and M. Bucciarelli, editors, *Proceedings of the Twenty-Seventh Annual Conference of the Cognitive Science Society*, pages 827–832. Erlbaum, Mahwah, NJ, 2005.

[9] T. L. Griffiths and M. L. Kalish. Language evolution by iterated learning with bayesian agents. *Cognitive Science*, 31:441–480, 2007.

[10] A. Gelman, J. B. Carlin, H. S. Stern, and D. B. Rubin. *Bayesian data analysis*. Chapman & Hall, New York, 1995.

[11] P. Hemmer and M. Steyvers. A bayesian account of reconstructive memory. In *Proceedings of the 30th Annual Conference of the Cognitive Science Society*, 2008.

[12] J. R. Norris. *Markov Chains*. Cambridge University Press, Cambridge, UK, 1997.

[13] D. H. Brainard. The Psychophysics Toolbox. *Spatial Vision*, 10:433–436, 1997.

[14] D. G. Pelli. The VideoToolbox software for visual psychophysics: Transforming numbers into movies. *Spatial Vision*, 10:437–442, 1997.
